# Locality Preserving Projections

**Xiaofei He**
Department of Computer Science
The University of Chicago
Chicago, IL 60637
`xiaofei@cs.uchicago.edu`

**Partha Niyogi**
Department of Computer Science
The University of Chicago
Chicago, IL 60637
`niyogi@cs.uchicago.edu`

## Abstract

Many problems in information processing involve some form of dimensionality reduction. In this paper, we introduce Locality Preserving Projections (LPP). These are linear projective maps that arise by solving a variational problem that optimally preserves the neighborhood structure of the data set. LPP should be seen as an alternative to Principal Component Analysis (PCA) – a classical linear technique that projects the data along the directions of maximal variance. When the high dimensional data lies on a low dimensional manifold embedded in the ambient space, the Locality Preserving Projections are obtained by finding the optimal linear approximations to the eigenfunctions of the Laplace Beltrami operator on the manifold. As a result, LPP shares many of the data representation properties of nonlinear techniques such as Laplacian Eigenmaps or Locally Linear Embedding. Yet LPP is linear and more crucially is defined everywhere in ambient space rather than just on the training data points. This is borne out by illustrative examples on some high dimensional data sets.

## 1. Introduction

Suppose we have a collection of data points of $n$-dimensional real vectors drawn from an unknown probability distribution. In increasingly many cases of interest in machine learning and data mining, one is confronted with the situation where $n$ is *very large*. However, there might be reason to suspect that the "intrinsic dimensionality" of the data is much lower. This leads one to consider methods of dimensionality reduction that allow one to represent the data in a lower dimensional space.

In this paper, we propose a new *linear* dimensionality reduction algorithm, called **Locality Preserving Projections** (LPP). It builds a graph incorporating neighborhood information of the data set. Using the notion of the Laplacian of the graph, we then compute a transformation matrix which maps the data points to a subspace. This linear transformation optimally preserves local neighborhood information in a certain sense. The representation map generated by the algorithm may be viewed as a linear discrete approximation to a continuous map that naturally arises from the geometry of the manifold [2]. The new algorithm is interesting from a number of perspectives.

1. The maps are designed to minimize a different objective criterion from the classical linear techniques.

2. The locality preserving quality of LPP is likely to be of particular use in informa-
tion retrieval applications. If one wishes to retrieve audio, video, text documents
under a vector space model, then one will ultimately need to do a nearest neighbor
search in the low dimensional space. Since LPP is designed for preserving local
structure, it is likely that a nearest neighbor search in the low dimensional space
will yield similar results to that in the high dimensional space. This makes for an
indexing scheme that would allow quick retrieval.

3. LPP is linear. This makes it fast and suitable for practical application. While a
number of non linear techniques have properties (1) and (2) above, we know of no
other linear projective technique that has such a property.

4. LPP is defined everywhere. Recall that nonlinear dimensionality reduction tech-
niques like ISOMAP[6], LLE[5], Laplacian eigenmaps[2] are defined only on the
training data points and it is unclear how to evaluate the map for new test points.
In contrast, the Locality Preserving Projection may be simply applied to any new
data point to locate it in the reduced representation space.

5. LPP may be conducted in the original space or in the reproducing kernel Hilbert
space(RKHS) into which data points are mapped. This gives rise to kernel LPP.

As a result of all these features, we expect the LPP based techniques to be a natural al-
ternative to PCA based techniques in exploratory data analysis, information retrieval, and
pattern classification applications.

## 2. Locality Preserving Projections

### 2.1. The linear dimensionality reduction problem

The generic problem of linear dimensionality reduction is the following. Given a set
$\mathbf{x}_1, \mathbf{x}_2, \cdots, \mathbf{x}_m$ in $\mathbf{R}^n$, find a transformation matrix $A$ that maps these $m$ points to a set
of points $\mathbf{y}_1, \mathbf{y}_2, \cdots, \mathbf{y}_m$ in $\mathbf{R}^l$ ($l \ll n$), such that $\mathbf{y}_i$ "represents" $\mathbf{x}_i$, where $\mathbf{y}_i = A^T \mathbf{x}_i$.
Our method is of particular applicability in the special case where $\mathbf{x}_1, \mathbf{x}_2, \cdots, \mathbf{x}_m \in \mathcal{M}$
and $\mathcal{M}$ is a nonlinear manifold embedded in $\mathbf{R}^n$.

### 2.2. The algorithm

Locality Preserving Projection (LPP) is a linear approximation of the nonlinear Laplacian
Eigenmap [2]. The algorithmic procedure is formally stated below:

1. **Constructing the adjacency graph**: Let $G$ denote a graph with $m$ nodes. We put
   an edge between nodes $i$ and $j$ if $\mathbf{x}_i$ and $\mathbf{x}_j$ are "close". There are two variations:

   (a) $\epsilon$-neighborhoods. [parameter $\epsilon \in \mathbf{R}$] Nodes $i$ and $j$ are connected by an edge
   if $\|\mathbf{x}_i - \mathbf{x}_j\|^2 < \epsilon$ where the norm is the usual Euclidean norm in $\mathbf{R}^n$.

   (b) $k$ nearest neighbors. [parameter $k \in \mathbf{N}$] Nodes $i$ and $j$ are connected by an
   edge if $i$ is among $k$ nearest neighbors of $j$ or $j$ is among $k$ nearest neighbors
   of $i$.

   *Note:* The method of constructing an adjacency graph outlined above is correct
   if the data actually lie on a low dimensional manifold. In general, however, one
   might take a more utilitarian perspective and construct an adjacency graph based
   on any principle (for example, perceptual similarity for natural signals, hyperlink
   structures for web documents, etc.). Once such an adjacency graph is obtained,
   LPP will try to optimally preserve it in choosing projections.

2. **Choosing the weights**: Here, as well, we have two variations for weighting the
   edges. $W$ is a sparse symmetric $m \times m$ matrix with $W_{ij}$ having the weight of the
   edge joining vertices $i$ and $j$, and 0 if there is no such edge.

(a) Heat kernel. [parameter $t \in \mathbf{R}$]. If nodes $i$ and $j$ are connected, put

$$W_{ij} = e^{-\frac{\|\mathbf{X}_i - \mathbf{X}_j\|^2}{t}}$$

The justification for this choice of weights can be traced back to [2].

(b) Simple-minded. [No parameter]. $W_{ij} = 1$ if and only if vertices $i$ and $j$ are connected by an edge.

3. **Eigenmaps**: Compute the eigenvectors and eigenvalues for the generalized eigenvector problem:

$$XLX^T\mathbf{a} = \lambda XDX^T\mathbf{a} \qquad (1)$$

where $D$ is a diagonal matrix whose entries are column (or row, since $W$ is symmetric) sums of $W$, $D_{ii} = \Sigma_j W_{ji}$. $L = D - W$ is the Laplacian matrix. The $i^{th}$ column of matrix $X$ is $\mathbf{x}_i$.

Let the column vectors $\mathbf{a}_0, \cdots, \mathbf{a}_{l-1}$ be the solutions of equation (1), ordered according to their eigenvalues, $\lambda_0 < \cdots < \lambda_{l-1}$. Thus, the embedding is as follows:

$$\mathbf{x}_i \rightarrow \mathbf{y}_i = A^T\mathbf{x}_i, A = (\mathbf{a}_0, \mathbf{a}_1, \cdots, \mathbf{a}_{l-1})$$

where $\mathbf{y}_i$ is a $l$-dimensional vector, and $A$ is a $n \times l$ matrix.

## 3. Justification

### 3.1. Optimal Linear Embedding

The following section is based on standard spectral graph theory. See [4] for a comprehensive reference and [2] for applications to data representation.

Recall that given a data set we construct a weighted graph $G = (V, E)$ with edges connecting nearby points to each other. Consider the problem of mapping the weighted graph $G$ to a line so that connected points stay as close together as possible. Let $\mathbf{y} = (y_1, y_2, \cdots, y_m)^T$ be such a map. A reasonable criterion for choosing a "good" map is to minimize the following objective function [2]

$$\sum_{ij}(y_i - y_j)^2 W_{ij}$$

under appropriate constraints. The objective function with our choice of $W_{ij}$ incurs a heavy penalty if neighboring points $\mathbf{x}_i$ and $\mathbf{x}_j$ are mapped far apart. Therefore, minimizing it is an attempt to ensure that if $\mathbf{x}_i$ and $\mathbf{x}_j$ are "close" then $y_i$ and $y_j$ are close as well.

Suppose $\mathbf{a}$ is a transformation vector, that is, $\mathbf{y}^T = \mathbf{a}^T X$, where the $i^{th}$ column vector of $X$ is $\mathbf{x}_i$. By simple algebra formulation, the objective function can be reduced to

$$\frac{1}{2}\sum_{ij}(y_i - y_j)^2 W_{ij} = \frac{1}{2}\sum_{ij}(\mathbf{a}^T\mathbf{x}_i - \mathbf{a}^T\mathbf{x}_j)^2 W_{ij}$$

$$= \sum_i \mathbf{a}^T\mathbf{x}_i D_{ii}\mathbf{x}_i^T\mathbf{a} - \sum_{ij}\mathbf{a}^T\mathbf{x}_i W_{ij}\mathbf{x}_j^T\mathbf{a} = \mathbf{a}^T X(D - W)X^T\mathbf{a} = \mathbf{a}^T XLX^T\mathbf{a}$$

where $X = [\mathbf{x}_1, \mathbf{x}_2, \cdots, \mathbf{x}_m]$, and $D$ is a diagonal matrix; its entries are column (or row, since $W$ is symmetric) sum of W, $D_{ii} = \Sigma_j W_{ij}$. $L = D - W$ is the Laplacian matrix [4]. Matrix $D$ provides a natural measure on the data points. The bigger the value $D_{ii}$ (corresponding to $y_i$) is, the more "important" is $y_i$. Therefore, we impose a constraint as follows:

$$\mathbf{y}^T D\mathbf{y} = 1 \Rightarrow \mathbf{a}^T XDX^T\mathbf{a} = 1$$

Finally, the minimization problem reduces to finding:

$$\arg\min_{\substack{\mathbf{a} \\ \mathbf{a}^T XDX^T\mathbf{a} = 1}} \mathbf{a}^T XLX^T\mathbf{a}$$

The transformation vector $\mathbf{a}$ that minimizes the objective function is given by the minimum eigenvalue solution to the generalized eigenvalue problem:

$$XLX^T\mathbf{a} = \lambda XDX^T\mathbf{a}$$

It is easy to show that the matrices $XLX^T$ and $XDX^T$ are symmetric and positive semi-definite. The vectors $\mathbf{a}_i(i = 0, 2, \cdots, l-1)$ that minimize the objective function are given by the minimum eigenvalue solutions to the generalized eigenvalue problem.

### 3.2. Geometrical Justification

The Laplacian matrix $L$ $(=D - W)$ for finite graph, or [4], is analogous to the Laplace Beltrami operator $\mathcal{L}$ on compact Riemannian manifolds. While the Laplace Beltrami operator for a manifold is generated by the Riemannian metric, for a graph it comes from the adjacency relation.

Let $\mathcal{M}$ be a smooth, compact, $d$-dimensional Riemannian manifold. If the manifold is embedded in $\mathbf{R}^n$ the Riemannian structure on the manifold is induced by the standard Riemannian structure on $\mathbf{R}^n$. We are looking here for a map from the manifold to the real line such that points close together on the manifold get mapped close together on the line. Let $f$ be such a map. Assume that $f : \mathcal{M} \to \mathbf{R}$ is twice differentiable.

Belkin and Niyogi [2] showed that the optimal map preserving locality can be found by solving the following optimization problem on the manifold:

$$\arg\min_{\|f\|_{L^2(\mathcal{M})}=1} \int_{\mathcal{M}} \|\nabla f\|^2$$

which is equivalent to [1]

$$\arg\min_{\|f\|_{L^2(\mathcal{M})}=1} \int_{\mathcal{M}} \mathcal{L}(f)f$$

where the integral is taken with respect to the standard measure on a Riemannian manifold. $\mathcal{L}$ is the Laplace Beltrami operator on the manifold, i.e. $\mathcal{L}f = -\operatorname{div} \nabla(f)$. Thus, the optimal $f$ has to be an eigenfunction of $\mathcal{L}$. The integral $\int_{\mathcal{M}} \mathcal{L}(f)f$ can be discretely approximated by $\langle f(X), Lf(X)\rangle = f^T(X)Lf(X)$ on a graph, where

$$f(X) = [f(\mathbf{x}_1), f(\mathbf{x}_2), \cdots, f(\mathbf{x}_m))]^T, f^T(X) = [f(\mathbf{x}_1), f(\mathbf{x}_2), \cdots, f(\mathbf{x}_m))]$$

If we restrict the map to be linear, i.e. $f(\mathbf{x}) = \mathbf{a}^T\mathbf{x}$, then we have

$$f(X) = X^T\mathbf{a} \Rightarrow \langle f(X), Lf(X)\rangle = f^T(X)Lf(X) = \mathbf{a}^T XLX^T\mathbf{a}$$

The constraint can be computed as follows,

$$\|f\|_{L^2(\mathcal{M})}^2 = \int_{\mathcal{M}} |f(\mathbf{x})|^2 d\mathbf{x} = \int_{\mathcal{M}} (\mathbf{a}^T\mathbf{x})^2 d\mathbf{x} = \int_{\mathcal{M}} (\mathbf{a}^T\mathbf{x}\mathbf{x}^T\mathbf{a})d\mathbf{x} = \mathbf{a}^T(\int_{\mathcal{M}} \mathbf{x}\mathbf{x}^T d\mathbf{x})\mathbf{a}$$

where $d\mathbf{x}$ is the standard measure on a Riemannian manifold. By spectral graph theory [4], the measure $d\mathbf{x}$ directly corresponds to the measure for the graph which is the degree of the vertex, i.e. $D_{ii}$. Thus, $\|f\|_{L^2(\mathcal{M})}^2$ can be discretely approximated as follows,

$$\|f\|_{L^2(\mathcal{M})}^2 = \mathbf{a}^T(\int_{\mathcal{M}} \mathbf{x}\mathbf{x}^T d\mathbf{x})\mathbf{a} \approx \mathbf{a}^T(\sum_i \mathbf{x}\mathbf{x}^T D_{ii})\mathbf{a} = \mathbf{a}^T XDX^T\mathbf{a}$$

Finally, we conclude that the optimal linear projective map, i.e. $f(\mathbf{x}) = \mathbf{a}^T\mathbf{x}$, can be obtained by solving the following objective function,

$$\arg\min_{\substack{\mathbf{a} \\ \mathbf{a}^T XDX^T\mathbf{a} = 1}} \mathbf{a}^T XLX^T\mathbf{a}$$

These projective maps are the optimal linear approximations to the eigenfunctions of the Laplace Beltrami operator on the manifold. Therefore, they are capable of discovering the nonlinear manifold structure.

### 3.3. Kernel LPP

Suppose that the Euclidean space $\mathbf{R}^n$ is mapped to a Hilbert space $\mathcal{H}$ through a nonlinear mapping function $\phi : \mathbf{R}^n \to \mathcal{H}$. Let $\phi(X)$ denote the data matrix in the Hilbert space, $\phi(X) = [\phi(\mathbf{x}_1), \phi(\mathbf{x}_2), \cdots, \phi(\mathbf{x}_m)]$. Now, the eigenvector problem in the Hilbert space can be written as follows:

$$[\phi(X)L\phi^T(X)]\nu = \lambda[\phi(X)D\phi^T(X)]\nu \tag{2}$$

To generalize LPP to the nonlinear case, we formulate it in a way that uses dot product exclusively. Therefore, we consider an expression of dot product on the Hilbert space $\mathcal{H}$ given by the following kernel function:

$$K(\mathbf{x}_i, \mathbf{x}_j) = (\phi(\mathbf{x}_i) \cdot \phi(\mathbf{x}_j)) = \phi^T(\mathbf{x}_i)\phi(\mathbf{x}_j)$$

Because the eigenvectors of (2) are linear combinations of $\phi(\mathbf{x}_1), \phi(\mathbf{x}_2), \cdots, \phi(\mathbf{x}_m)$, there exist coefficients $\alpha_i, i = 1, 2, \cdots, m$ such that

$$\nu = \sum_{i=1}^{m} \alpha_i \phi(\mathbf{x}_i) = \phi(X)\alpha$$

where $\alpha = [\alpha_1, \alpha_2, \cdots, \alpha_m]^T \in \mathbf{R}^m$.

By simple algebra formulation, we can finally obtain the following eigenvector problem:

$$KLK\alpha = \lambda KDK\alpha \tag{3}$$

Let the column vectors $\alpha^1, \alpha^2, \cdots, \alpha^m$ be the solutions of equation (3). For a test point $\mathbf{x}$, we compute projections onto the eigenvectors $\nu^k$ according to

$$(\nu^k \cdot \phi(\mathbf{x})) = \sum_{i=1}^{m} \alpha_i^k (\phi(\mathbf{x}) \cdot \phi(\mathbf{x}_i)) = \sum_{i=1}^{m} \alpha_i^k K(\mathbf{x}, \mathbf{x}_i)$$

where $\alpha_i^k$ is the $i^{th}$ element of the vector $\alpha^k$. For the original training points, the maps can be obtained by $\mathbf{y} = K\alpha$, where the $i^{th}$ element of $\mathbf{y}$ is the one-dimensional representation of $\mathbf{x}_i$. Furthermore, equation (3) can be reduced to

$$L\mathbf{y} = \lambda D\mathbf{y} \tag{4}$$

which is identical to the eigenvalue problem of Laplacian Eigenmaps [2]. This shows that Kernel LPP yields the same results as Laplacian Eigenmaps on the training points.

## 4. Experimental Results

In this section, we will discuss several applications of the LPP algorithm. We begin with two simple synthetic examples to give some intuition about how LPP works.

### 4.1. Simply Synthetic Example

Two simple synthetic examples are given in Figure 1. Both of the two data sets correspond essentially to a one-dimensional manifold. Projection of the data points onto the first basis would then correspond to a one-dimensional linear manifold representation. The second basis, shown as a short line segment in the figure, would be discarded in this low-dimensional example.

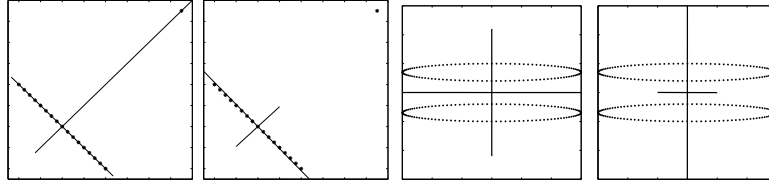

Figure 1: The first and third plots show the results of PCA. The second and forth plots show the results of LPP. The line segments describe the two bases. The first basis is shown as a longer line segment, and the second basis is shown as a shorter line segment. In this example, LPP is insensitive to the outlier and has more discriminating power than PCA.

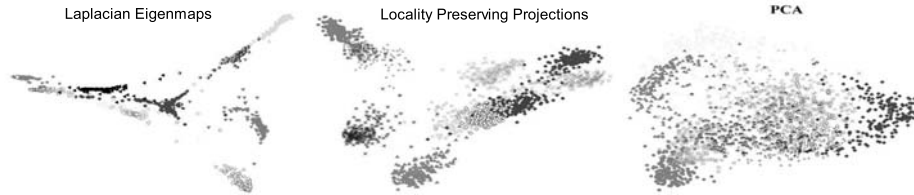

Figure 2: The handwritten digits ('0'-'9') are mapped into a 2-dimensional space. The left figure is a representation of the set of all images of digits using the Laplacian eigenmaps. The middle figure shows the results of LPP. The right figure shows the results of PCA. Each color corresponds to a digit.

LPP is derived by preserving local information, hence it is less sensitive to outliers than PCA. This can be clearly seen from Figure 1. LPP finds the principal direction along the data points at the left bottom corner, while PCA finds the principal direction on which the data points at the left bottom corner collapse into a single point. Moreover, LPP can has more discriminating power than PCA. As can be seen from Figure 1, the two circles are totally overlapped with each other in the principal direction obtained by PCA, while they are well separated in the principal direction obtained by LPP.

## 4.2. 2-D Data Visulization

An experiment was conducted with the *Multiple Features Database* [3]. This dataset consists of features of handwritten numbers ('0'-'9') extracted from a collection of Dutch utility maps. 200 patterns per class (for a total of 2,000 patterns) have been digitized in binary images. Digits are represented in terms of Fourier coefficients, profile correlations, Karhunen-Love coefficients, pixel average, Zernike moments and morphological features. Each image is represented by a 649-dimensional vector. These data points are mapped to a 2-dimensional space using different dimensionality reduction algorithms, PCA, LPP, and Laplacian Eigenmaps. The experimental results are shown in Figure 2. As can be seen, LPP performs much better than PCA. LPPs are obtained by finding the optimal linear approximations to the eigenfunctions of the Laplace Beltrami operator on the manifold. As a result, LPP shares many of the data representation properties of non linear techniques such as Laplacian Eigenmap. However, LPP is computationally much more tractable.

## 4.3. Manifold of Face Images

In this subsection, we applied the LPP to images of faces. The face image data set used here is the same as that used in [5]. This dataset contains 1965 face images taken from sequential frames of a small video. The size of each image is $20 \times 28$, with 256 gray levels

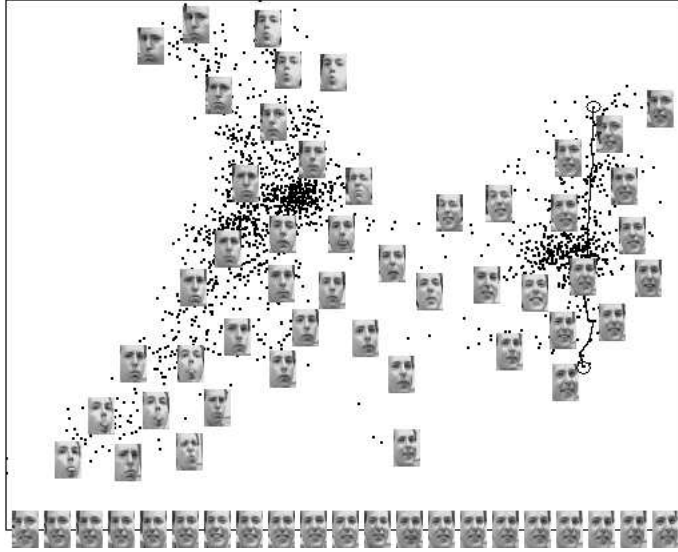

Figure 3: A two-dimensional representation of the set of all images of faces using the Locality Preserving Projection. Representative faces are shown next to the data points in different parts of the space. As can be seen, the facial expression and the viewing point of faces change smoothly.

Table 1: Face Recognition Results on Yale Database

|  | LPP | LDA | PCA |
|---|---|---|---|
| dims | 14 | 14 | 33 |
| error rate (%) | 16.0 | 20.0 | 25.3 |

per pixel. Thus, each face image is represented by a point in the 560-dimensional ambient space. Figure 3 shows the mapping results. The images of faces are mapped into the 2-dimensional plane described by the first two coordinates of the Locality Preserving Projections. It should be emphasized that the mapping from image space to low-dimensional space obtained by our method is linear, rather than nonlinear as in most previous work. The linear algorithm does detect the nonlinear manifold structure of images of faces to some extent. Some representative faces are shown next to the data points in different parts of the space. As can be seen, the images of faces are clearly divided into two parts. The left part are the faces with closed mouth, and the right part are the faces with open mouth. This is because that, by trying to preserve neighborhood structure in the embedding, the LPP algorithm implicitly emphasizes the natural clusters in the data. Specifically, it makes the neighboring points in the ambient space nearer in the reduced representation space, and faraway points in the ambient space farther in the reduced representation space. The bottom images correspond to points along the right path (linked by solid line), illustrating one particular mode of variability in pose.

### 4.4. Face Recognition

PCA and LDA are the two most widely used subspace learning techniques for face recognition [1][7]. These methods project the training sample faces to a low dimensional representation space where the recognition is carried out. The main supposition behind this procedure is that the face space (given by the feature vectors) has a lower dimension than the image space (given by the number of pixels in the image), and that the recognition of the faces can be performed in this reduced space. In this subsection, we consider the application of LPP to face recognition.

The database used for this experiment is the Yale face database [8]. It is constructed at the

Yale Center for Computational Vision and Control. It contains 165 grayscale images of 15 individuals. The images demonstrate variations in lighting condition (left-light, center-light, right-light), facial expression (normal, happy, sad, sleepy, surprised, and wink), and with/without glasses. Preprocessing to locate the the faces was applied. Original images were normalized (in scale and orientation) such that the two eyes were aligned at the same position. Then, the facial areas were cropped into the final images for matching. The size of each cropped image is $32 \times 32$ pixels, with 256 gray levels per pixel. Thus, each image can be represented by a 1024-dimensional vector.

For each individual, six images were taken with labels to form the training set. The rest of the database was considered to be the testing set. The training samples were used to learn a projection. The testing samples were then projected into the reduced space. Recognition was performed using a nearest neighbor classifier. In general, the performance of PCA, LDA and LPP varies with the number of dimensions. We show the best results obtained by them. The error rates are summarized in Table 1. As can be seen, LPP outperforms both PCA and LDA.

## 5. Conclusions

In this paper, we propose a new linear dimensionality reduction algorithm called Locality Preserving Projections. It is based on the same variational principle that gives rise to the Laplacian Eigenmap [2]. As a result it has similar locality preserving properties.

Our approach also has several possible advantages over recent nonparametric techniques for global nonlinear dimensionality reduction such as [2][5][6]. It yields a map which is simple, linear, and defined everywhere (and therefore on novel test data points). The algorithm can be easily kernelized yielding a natural non-linear extension.

Performance improvement of this method over Principal Component Analysis is demonstrated through several experiments. Though our method is a linear algorithm, it is capable of discovering the non-linear structure of the data manifold.

## Footnotes

[1]If $\mathcal{M}$ has a boundary, appropriate boundary conditions for $f$ need to be assumed.

## References

[1] P.N. Belhumeur, J.P. Hepanha, and D.J. Kriegman, "Eigenfaces vs. fisherfaces: recognition using class specific linear projection,"*IEEE. Trans. Pattern Analysis and Machine Intelligence*, vol. 19, no. 7, pp. 711-720, July 1997.

[2] M. Belkin and P. Niyogi, "Laplacian Eigenmaps and Spectral Techniques for Embedding and Clustering ," *Advances in Neural Information Processing Systems 14*, Vancouver, British Columbia, Canada, 2002.

[3] C. L. Blake and C. J. Merz, "UCI repository of machine learning databases", http://www.ics.uci.edu/ mlearn/MLRepository.html. Irvine, CA, University of California, Department of Information and Computer Science, 1998.

[4] Fan R. K. Chung, *Spectral Graph Theory,* Regional Conference Series in Mathematics, number 92, 1997.

[5] Sam Roweis, and Lawrence K. Saul, "Nonlinear Dimensionality Reduction by Locally Linear Embedding," *Science*, vol 290, 22 December 2000.

[6] Joshua B. Tenenbaum, Vin de Silva, and John C. Langford, "A Global Geometric Framework for Nonlinear Dimensionality Reduction," *Science*, vol 290, 22 December 2000.

[7] M. Turk and A. Pentland, "Eigenfaces for recognition," *Journal of Cognitive Neuroscience*, 3(1):71-86, 1991.

[8] Yale Univ. Face Database, http://cvc.yale.edu/projects/yalefaces/yalefaces.html.
